# The Bias-Variance Tradeoff and the Randomized GACV

**Grace Wahba, Xiwu Lin and Fangyu Gao**
Dept of Statistics
Univ of Wisconsin
1210 W Dayton Street
Madison, WI 53706
wahba,xiwu,fgao@stat.wisc.edu

**Dong Xiang**
SAS Institute, Inc.
SAS Campus Drive
Cary, NC 27513
sasdxx@unx.sas.com

**Ronald Klein, MD and Barbara Klein, MD**
Dept of Ophthalmalogy
610 North Walnut Street
Madison, WI 53706
kleinr,kleinb@epi.ophth.wisc.edu

## Abstract

We propose a new in-sample cross validation based method (randomized GACV) for choosing smoothing or bandwidth parameters that govern the bias-variance or fit-complexity tradeoff in 'soft' classification. Soft classification refers to a learning procedure which estimates the probability that an example with a given attribute vector is in class 1 *vs* class 0. The target for optimizing the the tradeoff is the Kullback-Liebler distance between the estimated probability distribution and the 'true' probability distribution, representing knowledge of an infinite population. The method uses a randomized estimate of the trace of a Hessian and mimics cross validation at the cost of a single relearning with perturbed outcome data.

## 1 INTRODUCTION

We propose and test a new in-sample cross-validation based method for optimizing the bias-variance tradeoff in 'soft classification' (Wahba *et al* 1994), called $ranGACV$ (randomized Generalized Approximate Cross Validation). Summarizing from Wahba *et al*(1994) we are given a training set consisting of $n$ examples, where for each example we have a vector $t \in \mathcal{T}$ of attribute values, and an outcome $y$, which is either 0 or 1. Based on the training data it is desired to estimate the probability $p$ of the outcome 1 for any new examples in the

future. In 'soft' classification the estimate $\hat{p}(t)$ of $p(t)$ is of particular interest, and might be used by a physician to tell patients how they might modify their risk $p$ by changing (some component of) $t$, for example, cholesterol as a risk factor for heart attack. Penalized likelihood estimates are obtained for $p$ by assuming that the logit $f(t), t \in \mathcal{T}$, which satisfies $p(t) = e^{f(t)}/(1 + e^{f(t)})$ is in some space $\mathcal{H}$ of functions. Technically $\mathcal{H}$ is a reproducing kernel Hilbert space, but you don't need to know what that is to read on. Let the training set be $\{y_i, t_i, i = 1, \cdots, n\}$. Letting $f_i = f(t_i)$, the negative log likelihood $\mathcal{L}\{y_i, t_i, f_i\}$ of the observations, given $f$ is

$$\mathcal{L}\{y_i, t_i, f_i\} = \sum_{i=1}^{n}[-y_i f_i + b(f_i)], \tag{1}$$

where $b(f) = log(1 + e^f)$. The penalized likelihood estimate of the function $f$ is the solution to: Find $f \in \mathcal{H}$ to minimize $I_\lambda(f)$:

$$I_\lambda(f) = \sum_{i=1}^{n}[-y_i f_i + b(f_i)] + J_\lambda(f), \tag{2}$$

where $J_\lambda(f)$ is a quadratic penalty functional depending on parameter(s) $\lambda = (\lambda_1, ..., \lambda_q)$ which govern the so called bias-variance tradeoff. Equivalently the components of $\lambda$ control the tradeoff between the complexity of $f$ and the fit to the training data. In this paper we sketch the derivation of the $ranGACV$ method for choosing $\lambda$, and present some preliminary but favorable simulation results, demonstrating its efficacy. This method is designed for use with penalized likelihood estimates, but it is clear that it can be used with a variety of other methods which contain bias-variance parameters to be chosen, and for which minimizing the Kullback-Liebler $(KL)$ distance is the target. In the work of which this is a part, we are concerned with $\lambda$ having multiple components. Thus, it will be highly convenient to have an in-sample method for selecting $\lambda$, if one that is accurate and computationally convenient can be found.

Let $p_\lambda$ be the the estimate and $p$ be the 'true' but unknown probability function and let $p_i = p(t_i), p_{\lambda i} = p_\lambda(t_i)$. For in-sample tuning, our criteria for a good choice of $\lambda$ is the $KL$ distance $KL(p, p_\lambda) = \frac{1}{n}\sum_{i=1}^{n}[p_i log\frac{p_i}{p_{\lambda i}} + (1 - p_i)log\frac{(1-p_i)}{(1-p_{\lambda i})}]$. We may replace $KL(p, p_\lambda)$ by the comparative $KL$ distance $(CKL)$, which differs from $KL$ by a quantity which does not depend on $\lambda$. Letting $f_{\lambda i} = f_\lambda(t_i)$, the $CKL$ is given by

$$CKL(p, p_\lambda) \equiv CKL(\lambda) = \frac{1}{n}\sum_{i=1}^{n}[-p_i f_{\lambda i} + b(f_{\lambda i})]. \tag{3}$$

$CKL(\lambda)$ depends on the unknown $p$, and it is desired is to have a good estimate or proxy for it, which can then be minimized with respect to $\lambda$.

It is known (Wong 1992) that no exact unbiased estimate of $CKL(\lambda)$ exists in this case, so that only approximate methods are possible. A number of authors have tackled this problem, including Utans and Moody(1993), Liu(1993), Gu(1992). The iterative $UBR$ method of Gu(1992) is included in GRKPACK (Wang 1997), which implements general smoothing spline ANOVA penalized likelihood estimates with multiple smoothing parameters. It has been successfully used in a number of practical problems, see, for example, Wahba *et al* (1994,1995). The present work represents an approach in the spirit of GRKPACK but which employs several approximations, and may be used with any data set, no matter how large, provided that an algorithm for solving the penalized likelihood equations, either exactly or approximately, can be implemented.

## 2   THE GACV ESTIMATE

In the general penalized likelihood problem the minimizer $f_\lambda(\cdot)$ of (2) has a representation

$$f_\lambda(t) = \sum_{\nu=1}^{M} d_\nu \phi_\nu(t) + \sum_{i=1}^{n} c_i Q_\lambda(t_i, t) \qquad (4)$$

where the $\phi_\nu$ span the null space of $J_\lambda$, $Q_\lambda(s,t)$ is a reproducing kernel (positive definite function) for the penalized part of $\mathcal{H}$, and $c = (c_1, \cdots, c_n)'$ satisfies $M$ linear conditions, so that there are (at most) $n$ free parameters in $f_\lambda$. Typically the unpenalized functions $\phi_\nu$ are low degree polynomials. Examples of $Q(t_i, \cdot)$ include radial basis functions and various kinds of splines; minor modifications include sigmoidal basis functions, tree basis functions and so on. See, for example Wahba(1990,1995), Girosi, Jones and Poggio(1995). If $f_\lambda(\cdot)$ is of the form (4) then $J_\lambda(f_\lambda)$ is a quadratic form in $c$. Substituting (4) into (2) results in $I_\lambda$ a convex functional in $c$ and $d$, and $c$ and $d$ are obtained numerically via a Newton Raphson iteration, subject to the conditions on $c$. For large $n$, the second sum on the right of (4) may be replaced by $\sum_{k=1}^{K} c_{i_k} Q_\lambda(t_{i_k}, t)$, where the $t_{i_k}$ are chosen via one of several principled methods.

To obtain the $GACV$ we begin with the ordinary leaving-out-one cross validation function $CV(\lambda)$ for the $CKL$:

$$CV(\lambda) = \frac{1}{n} \sum_{i=1}^{n} [-y_i f_{\lambda i}^{[-i]} + b(f_{\lambda i})], \qquad (5)$$

where $f_\lambda^{[-i]}$ the solution to the variational problem of (2) with the $i$th data point left out and $f_{\lambda i}^{[-i]}$ is the value of $f_\lambda^{[-i]}$ at $t_i$. Although $f_\lambda(\cdot)$ is computed by solving for $c$ and $d$ the $GACV$ is derived in terms of the values $(f_1, \cdots, f_n)'$ of $f$ at the $t_i$. Where there is no confusion between functions $f(\cdot)$ and vectors $(f_1, \cdots, f_n)'$ of values of $f$ at $t_1, \cdots, t_n$, we let $f = (f_1, \cdots, f_n)'$. For any $f(\cdot)$ of the form (4), $J_\lambda(f)$ also has a representation as a non-negative definite quadratic form in $(f_1, \cdots, f_n)'$. Letting $\Sigma_\lambda$ be twice the matrix of this quadratic form we can rewrite (2) as

$$I_\lambda(f, y) = \sum_{i=1}^{n} [-y_i f_i + b(f_i)] + \frac{1}{2} f' \Sigma_\lambda f. \qquad (6)$$

Let $W = W(f)$ be the $n \times n$ diagonal matrix with $\sigma_{ii} \equiv p_i(1 - p_i)$ in the $ii$th position. Using the fact that $\sigma_{ii}$ is the second derivative of $b(f_i)$, we have that $H = [W + \Sigma_\lambda]^{-1}$ is the inverse Hessian of the variational problem (6). In Xiang and Wahba (1996), several Taylor series approximations, along with a generalization of the leaving-out-one lemma (see Wahba 1990) are applied to (5) to obtain an approximate cross validation function $ACV(\lambda)$, which is a second order approximation to $CV(\lambda)$. Letting $h_{ii}$ be the $ii$th entry of $H$, the result is

$$CV(\lambda) \approx ACV(\lambda) = \frac{1}{n} \sum_{i=1}^{n} [-y_i f_{\lambda i} + b(f_{\lambda i})] + \frac{1}{n} \sum_{i=1}^{n} \frac{h_{ii} y_i (y_i - p_{\lambda i})}{[1 - h_{ii} \sigma_{ii}]} . \qquad (7)$$

Then the $GACV$ is obtained from the $ACV$ by replacing $h_{ii}$ by $\frac{1}{n} \sum_{i=1}^{n} h_{ii} \equiv \frac{1}{n} tr(H)$ and replacing $1 - h_{ii} \sigma_{ii}$ by $\frac{1}{n} tr[I - (W^{1/2} H W^{1/2})]$, giving

$$GACV(\lambda) = \frac{1}{n} \sum_{i=1}^{n} [-y_i f_{\lambda i} + b(f_{\lambda i})] + \frac{tr(H)}{n} \frac{\sum_{i=1}^{n} y_i (y_i - p_{\lambda i})}{tr[I - (W^{1/2} H W^{1/2})]} , \qquad (8)$$

where $W$ is evaluated at $f_\lambda$. Numerical results based on an exact calculation of (8) appear in Xiang and Wahba (1996). The exact calculation is limited to small $n$ however.

## 3 THE RANDOMIZED GACV ESTIMATE

Given any 'black box' which, given $\lambda$, and a training set $\{y_i, t_i\}$ produces $f_\lambda(\cdot)$ as the minimizer of (2), and thence $f_\lambda = (f_{\lambda 1}, \cdots, f_{\lambda n})'$, we can produce randomized estimates of $trH$ and $tr[I - W^{1/2}HW^{1/2}]$ without having any explicit calculations of these matrices. This is done by running the 'black box' on perturbed data $\{y_i + \delta_i, t_i\}$. For the $y_i$ Gaussian, randomized trace estimates of the Hessian of the variational problem (the 'influence matrix') have been studied extensively and shown to be essentially as good as exact calculations for large $n$, see for example Girard(1998). Randomized trace estimates are based on the fact that if $A$ is any square matrix and $\delta$ is a zero mean random $n$-vector with independent components with variance $\sigma_\delta^2$, then $E\delta'A\delta = \frac{1}{\sigma_\delta^2}trA$. See Gong et al(1998) and references cited there for experimental results with multiple regularization parameters. Returning to the 0-1 data case, it is easy to see that the minimizer $f_\lambda(\cdot)$ of $I_\lambda$ is continuous in $y$, not withstanding the fact that in our training set the $y_i$ take on only values 0 or 1. Letting $f_\lambda^y = (f_{\lambda 1}, \cdots, f_{\lambda n})'$ be the minimizer of (6) given $y = (y_1, \cdots, y_n)'$, and $f_\lambda^{y+\delta}$ be the minimizer given data $y+\delta = (y_1+\delta_1, \cdots, y_n+\delta_n)'$ (the $t_i$ remain fixed), Xiang and Wahba (1997) show, again using Taylor series expansions, that $f_\lambda^{y+\delta} - f_\lambda^y \sim [W(f_\lambda^y) + \Sigma_\lambda]^{-1}\delta$. This suggests that $\frac{1}{\sigma_\delta^2}\delta'(f_\lambda^{y+\delta} - f_\lambda^y)$ provides an estimate of $tr[W(f_\lambda^y) + \Sigma_\lambda]^{-1}$. However, if we take the solution $f_\lambda^y$ to the nonlinear system for the original data $y$ as the initial value for a Newton-Raphson calculation of $f_\lambda^{y+\delta}$ things become even simpler. Applying a one step Newton-Raphson iteration gives

$$f_\lambda^{y+\delta,1} = f_\lambda^y - [\frac{\partial^2 I_\lambda}{\partial f'\partial f}(f_\lambda^y, y+\delta)]^{-1}\frac{\partial I_\lambda}{\partial f}(f_\lambda^y, y+\delta). \tag{9}$$

Since $\frac{\partial I_\lambda}{\partial f}(f_\lambda^y, y+\delta) = -\delta + \frac{\partial I_\lambda}{\partial f}(f_\lambda^y, y) = -\delta$, and $[\frac{\partial^2 I_\lambda}{\partial f'\partial f}(f_\lambda^y, y+\delta)]^{-1} = [\frac{\partial^2 I_\lambda}{\partial f'\partial f}(f_\lambda^y, y)]^{-1}$, we have $f_\lambda^{y+\delta,1} = f_\lambda^y + [\frac{\partial^2 I_\lambda}{\partial f'\partial f}(f_\lambda^y, y)]^{-1}\delta$ so that $f_\lambda^{y+\delta,1} - f_\lambda^y = [W(f_\lambda^y) + \Sigma_\lambda]^{-1}\delta$. The result is the following $ranGACV$ function:

$$ranGACV(\lambda) = \frac{1}{n}\sum_{i=1}^n [-y_i f_{\lambda i} + b(f_{\lambda i})] + \frac{\delta'(f_\lambda^{y+\delta,1} - f_\lambda^y)}{n}\frac{\sum_{i=1}^n y_i(y_i - p_{\lambda i})}{[\delta'\delta - \delta'W(f_\lambda^y)(f_\lambda^{y+\delta,1} - f_\lambda^y)]}. \tag{10}$$

To reduce the variance in the term after the '+' in (10), we may draw $R$ independent replicate vectors $\delta_1, \cdots, \delta_R$, and replace the term after the '+' in (10)by $\frac{1}{R}\sum_{r=1}^R \frac{\delta_r'(f_\lambda^{y+\delta_r,1} - f_\lambda^y)}{n}\frac{\sum_{i=1}^n y_i(y_i - p_{\lambda i})}{[\delta_r'\delta_r - \delta_r'W(f_\lambda^y)(f_\lambda^{y+\delta_r,1} - f_\lambda^y)]}$ to obtain an $R$-replicated $ranGACV(\lambda)$ function.

## 4 NUMERICAL RESULTS

In this section we present simulation results which are representative of more extensive simulations to appear elsewhere. In each case, $K << n$ was chosen by a sequential clustering algorithm. In that case, the $t_i$ were grouped into $K$ clusters and one member of each cluster selected at random. The model is fit. Then the number of clusters is doubled and the model is fit again. This procedure continues until the fit does not change. In the randomized trace estimates the random variates were Gaussian. Penalty functionals were (multivariate generalizations of) the cubic spline penalty functional $\lambda \int_0^1 (f''(x))^2$, and smoothing spline ANOVA models were fit.

## 4.1   EXPERIMENT 1. SINGLE SMOOTHING PARAMETER

In this experiment $t \in [0,1]$, $f(t) = 2sin(10t)$, $t_i = (i - .5)/500, i = 1, \cdots, 500$. A random number generator produced 'observations' $y_i = 1$ with probability $p_i = e^{f_i}/(1 + e^{f_i})$, to get the training set. $Q_\lambda$ is given in Wahba(1990) for this cubic spline case, $K = 50$. Since the true $p$ is known, the true $CKL$ can be computed. Fig. 1(a) gives a plot of $CKL(\lambda)$ and 10 replicates of $ranGACV(\lambda)$. In each replicate $R$ was taken as 1, and $\delta$ was generated anew as a Gaussian random vector with $\sigma_\delta = .001$. Extensive simulations with different $\sigma_\delta$ showed that the results were insensitive to $\sigma_\delta$ from 1.0 to $10^{-6}$. The minimizer of $CKL$ is at the filled-in circle and the 10 minimizers of the 10 replicates of $ranGACV$ are the open circles. Any one of these 10 provides a rather good estimate of the $\lambda$ that goes with the filled-in circle. Fig. 1(b) gives the same experiment, except that this time $R = 5$. It can be seen that the minimizers $ranGACV$ become even more reliable estimates of the minimizer of $CKL$, and the $CKL$ at all of the $ranGACV$ estimates are actually quite close to its minimum value.

## 4.2   EXPERIMENT 2. ADDITIVE MODEL WITH $\lambda = (\lambda_1, \lambda_2)$

Here $t \in [0,1] \otimes [0,1]$. $n = 500$ values of $t_i$ were generated randomly according to a uniform distribution on the unit square and the $y_i$ were generated according to $p_i = e^{f_i}/(1 + e^{f_i})$ with $t = (x_1, x_2)$ and $f(t) = 5\sin 2\pi x_1 - 3sin2\pi x_2$. An additive model as a special case of the smoothing spline ANOVA model (see Wahba *et al*, 1995), of the form $f(t) = \mu + f_1(x_1) + f_2(x_2)$ with cubic spline penalties on $f_1$ and $f_2$ were used. $K = 50, \sigma_\delta = .001, R = 5$. Figure 1(c) gives a plot of $CKL(\lambda_1, \lambda_2)$ and Figure 1(d) gives a plot of $ranGACV(\lambda_1, \lambda_2)$. The open circles mark the minimizer of $ranGACV$ in both plots and the filled in circle marks the minimizer of $CKL$. The inefficiency, as measured by $CKL(\hat{\lambda})/min_\lambda CKL(\lambda)$ is 1.01. Inefficiencies near 1 are typical of our other similar simulations.

## 4.3   EXPERIMENT 3. COMPARISON OF ranGACV AND UBR

This experiment used a model similar to the model fit by GRKPACK for the risk of progression of diabetic retinopathy given $t = (x_1, x_2, x_3) =$ (duration, glycosylated hemoglobin, body mass index) in Wahba *et al*(1995) as 'truth'. A training set of 669 examples was generated according to that model, which had the structure $f(x_1, x_2, x_3) = \mu + f_1(x_1) + f_2(x_2) + f_3(x_3) + f_{1,3}(x_1, x_3)$. This (synthetic) training set was fit by GRK-PACK and also using $K = 50$ basis functions with $ranGACV$. Here there are $p = 6$ smoothing parameters (there are 3 smoothing parameters in $f_{13}$) and the $ranGACV$ function was searched by a downhill simplex method to find its minimizer. Since the 'truth' is known, the $CKL$ for $\hat{\lambda}$ and for the GRKPACK fit using the iterative $UBR$ method were computed. This was repeated 100 times, and the 100 pairs of $CKL$ values appears in Figure 1(e). It can be seen that the $UBR$ and $ranGACV$ give similar $CKL$ values about 90% of the time, while the $ranGACV$ has lower $CKL$ for most of the remaining cases.

## 4.4   DATA ANALYSIS: AN APPLICATION

Figure 1(f) represents part of the results of a study of association at baseline of pigmentary abnormalities with various risk factors in 2585 women between the ages of 43 and 86 in the Beaver Dam Eye Study, R. Klein *et al*(1995). The attributes are: $x_1 =$ age, $x_2 =$body mass index, $x_3 =$ systolic blood pressure, $x_4 =$ cholesterol. $x_5$ and $x_6$ are indicator variables for taking hormones, and history of drinking. The smoothing spline ANOVA model fitted was $f(t) = \mu + d_1 x_1 + d_2 x_2 + f_3(x_3) + f_4(x_4) + f_{34}(x_3, x_4) + d_5 I(x_5) + d_6 I(x_6)$, where $I$ is the indicator function. Figure 1(e) represents a cross section of the fit for $x_5 = no, x_6 = no$,

$x_2, x_3$ fixed at their medians and $x_1$ fixed at the 75th percentile. The dotted lines are the Bayesian confidence intervals, see Wahba *et al*(1995). There is a suggestion of a borderline inverse association of cholesterol. The reason for this association is uncertain. More details will appear elsewhere.

Principled soft classification procedures can now be implemented in much larger data sets than previously possible, and the $ranGACV$ should be applicable in general learning.

## References

Girard, D. (1998), 'Asymptotic comparison of (partial) cross-validation, GCV and randomized GCV in nonparametric regression', *Ann. Statist.* **126**, 315–334.

Girosi, F., Jones, M. & Poggio, T. (1995), 'Regularization theory and neural networks architectures', *Neural Computation* **7**, 219–269.

Gong, J., Wahba, G., Johnson, D. & Tribbia, J. (1998), 'Adaptive tuning of numerical weather prediction models: simultaneous estimation of weighting, smoothing and physical parameters', *Monthly Weather Review* **125**, 210–231.

Gu, C. (1992), 'Penalized likelihood regression: a Bayesian analysis', *Statistica Sinica* **2**, 255–264.

Klein, R., Klein, B. & Moss, S. (1995), 'Age-related eye disease and survival. the Beaver Dam Eye Study', *Arch Ophthalmol* **113**, 1995.

Liu, Y. (1993), Unbiased estimate of generalization error and model selection in neural network, manuscript, Department of Physics, Institute of Brain and Neural Systems, Brown University.

Utans, J. & Moody, J. (1993), Selecting neural network architectures via the prediction risk: application to corporate bond rating prediction, *in* 'Proc. First Int'l Conf. on Artificial Intelligence Applications on Wall Street', IEEE Computer Society Press.

Wahba, G. (1990), *Spline Models for Observational Data*, SIAM. CBMS-NSF Regional Conference Series in Applied Mathematics, v. 59.

Wahba, G. (1995), Generalization and regularization in nonlinear learning systems, *in* M. Arbib, ed., 'Handbook of Brain Theory and Neural Networks', MIT Press, pp. 426–430.

Wahba, G., Wang, Y., Gu, C., Klein, R. & Klein, B. (1994), Structured machine learning for 'soft' classification with smoothing spline ANOVA and stacked tuning, testing and evaluation, *in* J. Cowan, G. Tesauro & J. Alspector, eds, 'Advances in Neural Information Processing Systems 6', Morgan Kauffman, pp. 415–422.

Wahba, G., Wang, Y., Gu, C., Klein, R. & Klein, B. (1995), 'Smoothing spline ANOVA for exponential families, with application to the Wisconsin Epidemiological Study of Diabetic Retinopathy', *Ann. Statist.* **23**, 1865–1895.

Wang, Y. (1997), 'GRKPACK: Fitting smoothing spline analysis of variance models to data from exponential families', *Commun. Statist. Sim. Comp.* **26**, 765–782.

Wong, W. (1992), Estimation of the loss of an estimate, Technical Report 356, Dept. of Statistics, University of Chicago, Chicago, Il.

Xiang, D. & Wahba, G. (1996), 'A generalized approximate cross validation for smoothing splines with non-Gaussian data', *Statistica Sinica* **6**, 675–692, preprint TR 930 available via www.stat.wisc.edu/~wahba − > TRLIST.

Xiang, D. & Wahba, G. (1997), Approximate smoothing spline methods for large data sets in the binary case, Technical Report 982, Department of Statistics, University of Wisconsin, Madison WI. To appear in the Proceedings of the 1997 ASA Joint Statistical Meetings, Biometrics Section, pp 94-98 (1998). Also in TRLIST as above.

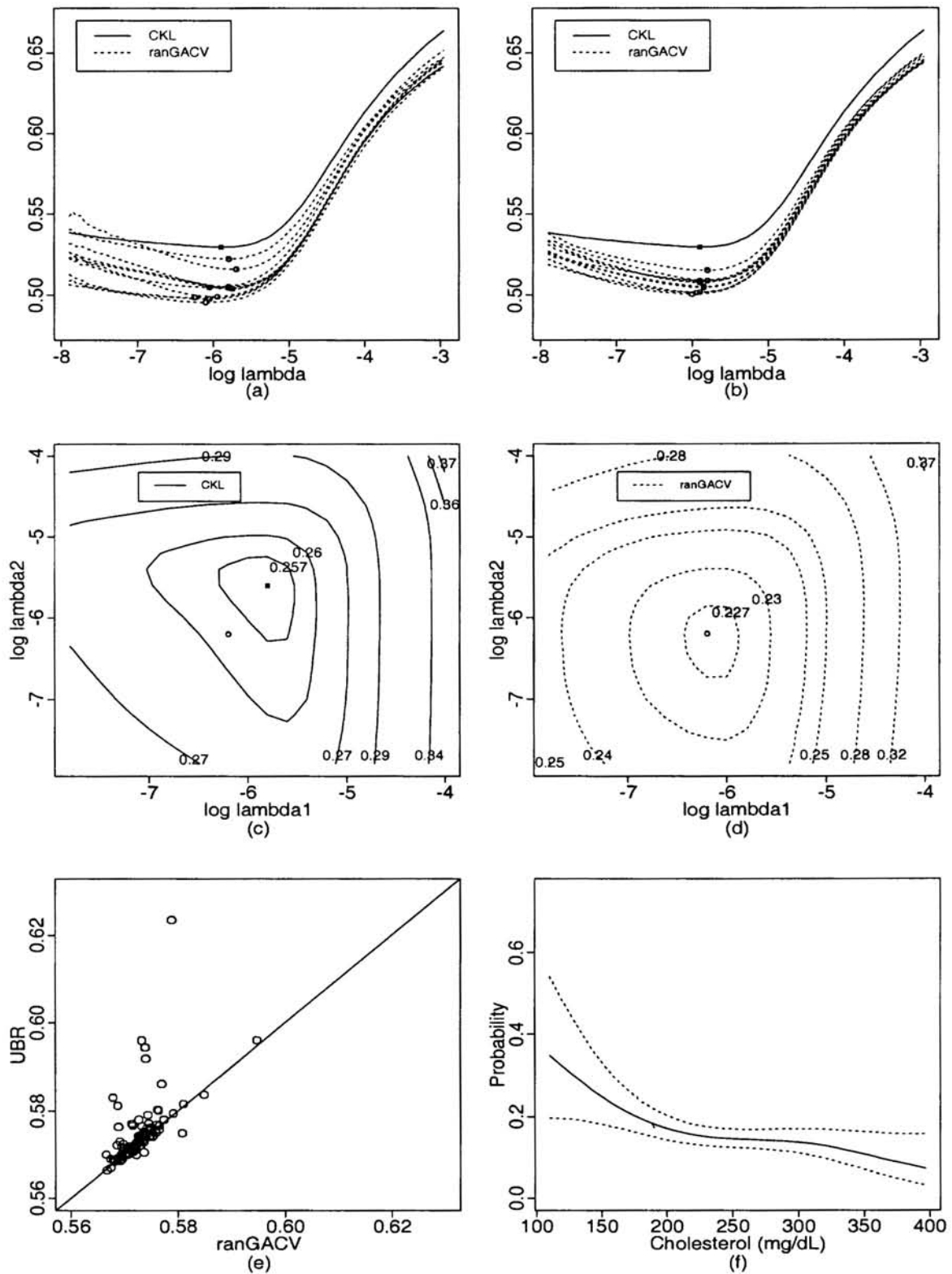

Figure 1: (a) and (b): Single smoothing parameter comparison of $ranGACV$ and $CKL$. (c) and (d): Two smoothing parameter comparison of $ranGACV$ and $CKL$. (e): Comparison of $ranGACV$ and $UBR$. (f): Probability estimate from Beaver Dam Study